# Gaussian process modulated renewal processes

**Vinayak Rao**
Gatsby Computational Neuroscience Unit
University College London
vrao@gatsby.ucl.ac.uk

**Yee Whye Teh**
Gatsby Computational Neuroscience Unit
University College London
ywteh@gatsby.ucl.ac.uk

## Abstract

Renewal processes are generalizations of the Poisson process on the real line whose intervals are drawn i.i.d. from some distribution. Modulated renewal processes allow these interevent distributions to vary with time, allowing the introduction of nonstationarity. In this work, we take a nonparametric Bayesian approach, modelling this nonstationarity with a Gaussian process. Our approach is based on the idea of *uniformization*, which allows us to draw exact samples from an otherwise intractable distribution. We develop a novel and efficient MCMC sampler for posterior inference. In our experiments, we test these on a number of synthetic and real datasets.

## 1 Introduction

Renewal processes are stochastic point processes on the real line where intervals between successive points (times) are drawn i.i.d. from some distribution. The simplest example of a renewal process is the homogeneous Poisson process, whose interevent times are exponentially distributed. A limitation of this is the memoryless property of the exponential distribution, resulting in an 'as bad as old after a repair' property [1] that is not true of many real-world phenomena. For example, immediately after firing, a neuron is depleted of its resources and incapable of firing again, and the gamma distribution is used to model interspike intervals [2]. Similarly, because of the phenomenon of elastic rebound, some time is required to recharge stresses released after an earthquake and an inverse Gaussian distribution is used to model intervals between major earthquakes [3]. Other examples include using the Pareto distribution to better capture the burstiness and self-similarity of network traffic arrival times [4], and the Erlang distribution to model the fact that buying incidence of frequently purchased goods is less variable than Poisson [5].

Modelling interevent times as i.i.d. draws from a general renewal density can allow larger or smaller variances than an exponential with the same mean (overdispersion or underdispersion), but effectively encodes an 'as good as new after a repair' property. Again, this is often only an approximation: because of age or other time-varying factors, the interevent distribution of the point process may vary with time. For instance, internet traffic can vary with time of the day, day of the week and in response to advertising and seasonal trends. Similarly, an external stimulus can modulate the firing rate of the neuron, economic trends can modulate financial transactions etc. The most popular way of modelling this nonstationarity is via an inhomogeneous Poisson process whose intensity function determines the instantaneous event rate, and there has also been substantial work extending this to renewal processes in various ways (see section 2.2).

In this paper, we describe a nonparametric Bayesian approach where a renewal process is modulated by a random intensity function which is given a Gaussian process prior. Our approach extends work by [6] on the Poisson process, using a generalization of the idea of Poisson thinning called *uniformization* [7] to draw exact samples from the model. We extend recent ideas from [8] to develop a more natural and efficient block Gibbs sampler than the incremental Metropolis-Hastings algorithm used in [6]. In our experiments we demonstrate the usefulness of our model and sampler on a number of synthetic and real-world datasets.

## 2 Modulated renewal processes

Consider a renewal process $\mathcal{R}$ over an interval $[0, T]$ whose interevent time is distributed according to a *renewal density* $g$. Let $G = \{G_1, G_2, ...\}$ be the ordered set of event times sampled from this renewal process, i.e.

$$G \sim \mathcal{R}(g) \tag{1}$$

For simplicity[1] we place a starting event $G_0$ at time 0, so for each $i \geq 1$ we have $(G_i - G_{i-1}) \sim g$.

Associated with the renewal density $g$ is a *hazard function* $h$, where $h(\tau)\Delta$, for infinitesimal $\Delta > 0$, is the probability of the interevent interval being in $[\tau, \tau + \Delta]$ conditioned on it being at least $\tau$, i.e.

$$h(\tau) = \frac{g(\tau)}{1 - \int_0^\tau g(u)du} \tag{2}$$

Let $\lambda(t)$ be some time-varying intensity function. A simple way to introduce nonstationarity into a renewal process is to modulate the hazard function by $\lambda(t)$ so that it depends on both the time $\tau$ since the last event, and on the absolute time $t$ [9, 10]:

$$h(\tau, t) \equiv m(h(\tau), \lambda(t)) \tag{3}$$

where $m(\cdot, \cdot)$ is some *interaction function*. Examples include additive $(h(\tau) + \lambda(t))$ and multiplicative $(h(\tau)\lambda(t))$ interactions. For concreteness, we assume multiplicative interactions in what follows, however our results extend easily to general interaction functions.

With a modulated hazard rate, the distribution of interevent times is no longer stationary. Instead, plugging a multiplicative interaction into (2) and solving for $g$ (see the supplementary material for details), we get

$$g(\tau|t_{prev}) = \lambda(t_{prev} + \tau)h(\tau) \exp\left(-\int_0^\tau \lambda(t_{prev} + u)h(u)du\right) \tag{4}$$

where $t_{prev}$ is the previous event time. Observe that equation (4) encompasses the inhomogeneous Poisson process as a special case (a constant hazard function with multiplicative modulation).

### 2.1 Gaussian process intensity functions

In this paper we are interested in estimating both parameters of the hazard function $h(\tau)$ as well as the intensity function $\lambda(t)$ itself. Taking a Bayesian nonparametric approach, we model $\lambda(t)$ using a Gaussian process (GP) [11] prior, which has support over a rich class of functions and offers a flexibility not afforded by parametric approaches. We call the resulting model a *Gaussian process modulated renewal process*. A minor issue is that samples from a GP can take negative values; we address this using a sigmoidal link function. Finally, we use a gamma family for the hazard function: $h(\tau) = \frac{x^{\gamma-1}e^{-\gamma x}}{\int_x^\infty u^{\gamma-1}e^{-\gamma u}du}$ where $\gamma$ is the shape parameter[2]. Our complete model is thus

$$l(\cdot) \sim \mathcal{GP}(\mu, K), \qquad \lambda(\cdot) = \lambda^* \sigma(l(\cdot)), \qquad G \sim \mathcal{R}(\lambda(\cdot), h(\cdot)) \tag{5}$$

where $\mu$ and $K$ are the GP mean and covariance kernel, $\lambda^*$ is a positive scale parameter, and $\sigma(x) = (1 + \exp(-x))^{-1}$. We place a gamma hyperprior on $\lambda^*$ as well as hyperpriors on the GP hyperparameters.

### 2.2 Related work

The idea of defining a nonstationary renewal process by modulating the hazard function dates back to Cox [9]. Early work [12] focussed on hypothesis testing for the stationarity assumption. [13, 14, 1] proposed parametric (generalized linear) models where the intensity function was a linear combination of some known functions; these regression coefficients were estimated via maximum likelihood. [15] considers general modulated hazard functions as well; however they assume it has known form and are concerned with calculating statistical properties of the resulting process.

Finally, [10] describe a model that is a generalization of ours, but again have to resort to maximum likelihood estimation (our ideas can easily be extended to their more general model too).

A different approach to producing inhomogeneity is by first sampling from a homogeneous renewal process and then rescaling time [16, 17]. The trend renewal process [18] uses such an approach, and the authors propose an iterative kernel smoothing scheme to approximate a maximum likelihood estimate of the intensity function. [2] uses time-rescaling to introduce inhomogeneity and, similar to us, a Gaussian process prior for the intensity function. Unlike us, they had to discretize time and used a variational approach to inference.

Finally, we note that our approach generalizes [6], who describe a doubly stochastic *Poisson* process and an MCMC sampler which does not require time discretization. In the next sections we describe a generalization of their model to the inhomogeneous renewal process using a twist on a classical idea called *uniformization*.

## 3   Sampling via Uniformization

Before we consider Markov chain Monte Carlo (MCMC) inference for our model, observe that to even naïvely generate samples from the prior is difficult; this requires evaluating integrals of a continuous-time function drawn from a GP (see equation (4)). One approach is to evaluate these integrals numerically by discretizing time [2], which can be time consuming and introduce approximation errors. In section 3.2 we will show how a classical idea called *uniformization* allows us to efficiently draw *exact* samples from the model, without approximations due to discretization. Then in section 4 we will develop a novel MCMC algorithm based on uniformization.

### 3.1   Modulated Poisson processes

We start with thinning, a well-known result to sample from an inhomogeneous Poisson process with intensity $\lambda(t)$. Suppose that $\lambda(t)$ is upper bounded by some constant $\Omega$. Let $E$ be a set of locations sampled from a *homogeneous* Poisson process with rate $\Omega$. We *thin* this set by deleting each point $e \in E$ independently with probability $1 - \frac{\lambda(e)}{\Omega}$. Let $F$ be the remaining set of points. Then:

**Proposition 1** ([19]). *The set $F$ is a draw from a Poisson process with intensity function $\lambda(t)$ .*

### 3.2   Modulated renewal processes

Less well-known is a generalization of this result to renewal processes [13]. Note that the thinning result of the previous section builds on the memoryless property of the exponential distribution (or the *complete randomness* [20] of the Poisson process): events in disjoint sets occur independently of each other. For a renewal process, events are no longer independent of their neighbours. This suggests a generalization of thinning involving a *Markov chain* over the set of events. This idea of thinning a Poisson process by a *subordinated* Markov chain is called uniformization [7].

[21] describes a uniformization scheme to sample from a *homogeneous* renewal process. We extend it to the modulated case here. We will assume that both the intensity function $\lambda(t)$ and the hazard function $h(\tau)$ are bounded, so that there exists a constant $\Omega$ such that

$$\Omega \geq \max_{t,\tau} h(\tau)\lambda(t) \tag{6}$$

Note that because of the sigmoidal link function, our model has $\lambda(t) \leq \lambda^*$, while the gamma hazard $h(\tau)$ is bounded by the shape parameter $\gamma$ if $\gamma \geq 1$. We now sample a set of times $E = \{E_0 = 0, E_1, E_2, \ldots\}$ from a homogeneous Poisson process with rate $\Omega$ and thin this set by running a discrete time Markov chain on the times in $E$. Let $Y_0 = 0, Y_1, Y_2, \ldots$ be an integer-valued Markov chain, where each $Y_i$ either equals $Y_{i-1}$ or $i$. We interpret $Y_i$ as indicating the index of the last *unthinned* event prior or equal to $E_i$. That is, $Y_i = Y_{i-1}$ means that $E_i$ is thinned, and $Y_i = i$ means $E_i$ is not thinned. Note that $E_i - E_{Y_i}$ gives the time since the last unthinned event. For $i > j \geq 0$, define the transition probabilities of the Markov chain (conditioned on $E$) as follows,

$$p(Y_i = i | Y_{i-1} = j) = \frac{h(E_i - E_j)\lambda(E_i)}{\Omega}, \quad p(Y_i = j | Y_{i-1} = j) = 1 - \frac{h(E_i - E_j)\lambda(E_i)}{\Omega} \tag{7}$$

After drawing a sample from $Y$, we define $F = \{E_i \in E \text{ s.t. } Y_i = i\}$.

**Proposition 2.** *For any $\Omega \geq \max_{t,\tau} h(\tau)\lambda(t)$, $F$ is a sample from a modulated renewal process with hazard $h(\cdot)$ and modulating intensity $\lambda(\cdot)$.*

The proof of this is included in the supplementary material. The basic idea is to write down the probability $p(E, Y)$ of the whole generative process and marginalize out the thinned times, showing that the resulting interevent time is simply (4). For a different proof of a similar result, see [13].

Now recall that we have a GP prior for $l(\cdot)$. The uniformization procedure above only requires the intensity function evaluated at the times in $E$ (which is finite on a finite interval), and this is easily obtained by sampling from a finite dimensional Gaussian $\mathcal{N}(\mu_E, K_E)$, with mean and covariance being the corresponding GP parameters $\mu$ and $K$ evaluated at $E$. Our procedure to sample from a GP-modulated renewal process now follows: sample from a homogeneous Poisson process $\mathcal{P}(\Omega)$ on $[0, T]$, instantiate the GP on this finite set of points and then thin the set by running the Markov chain described previously. Defining $l_E$ as $l(t)$ evaluated on the set $E$, $E_i^*$ as the restriction of $E$ to the interval $(F_{i-1}, F_i)$, and $F_{i+1} = T$ we can write the joint distribution:

$$P(F, l, E) = \Omega^{|E|} e^{-\Omega T} \mathcal{N}(l_E | \mu_E, K_E) \prod_{i=1}^{|F|} \left( \frac{\lambda(F_i) h(F_i - F_{i-1})}{\Omega} \right) \prod_{i=1}^{|F|+1} \prod_{e \in E_i^*} \left( 1 - \frac{\lambda(e) h(e - F_{i-1})}{\Omega} \right) \quad (8)$$

## 4 Inference

We now consider posterior inference on the modulating function $\lambda(t)$ (and any unknown hyperparameters) given an observed set of event times $G$. Our sampling algorithm is based on ideas developed in [8]. We imagine $G$ was generated via uniformization, so that there exists an unobserved set of thinned events $\tilde{G}$. We then proceed by Markov chain Monte Carlo, setting up a Markov chain whose state consists of the number and locations of $\tilde{G}$, the values of the GP on the set $G \cup \tilde{G}$ as well as the current sampled hyperparameters. Note from equation (8) that given these values, the value of the modulating function at any other location is independent of the observations and can be sampled from the conditional distribution of a multivariate Gaussian.

The challenge now is to construct a transition operator that results in this Markov chain having the desired posterior distribution as its equilibrium distribution. In their work, [6] defined a transition operator by proposing insertions and deletions of thinned events as well as by perturbing their locations. The proposals were accepted or rejected using a Metropolis-Hastings correction. The remaining variables were updated using standard Gaussian process techniques. We show below that instead of incrementally updating $\tilde{G}$, it is actually possible to produce a new *independent* sample of the entire set $\tilde{G}$ (conditioned on all other variables). This leads to a more natural sampler that does not require any external tuning and that mixes more rapidly.

To understand our algorithm, suppose first that the modulating function $\lambda(t)$ is known for all $t$. Then, from (4), the probability of the set of events $G$ on the interval $[0, T]$ is[3]:

$$P(G | \lambda(t)) = \prod_{i=1}^{|G|} \lambda(G_i) h(G_i - G_{i-1}) \prod_{i=1}^{|G|+1} \exp\left( - \int_{G_{i-1}}^{G_i} \lambda(t) h(t - G_{i-1}) dt \right) \quad (9)$$

Now, suppose that in each consecutive interval $(G_{i-1}, G_i)$ we independently sample a set of events $\tilde{G}_i^*$ from an inhomogeneous Poisson process with rate $(\Omega - \lambda(t) h(t - G_{i-1}))$, and let $\tilde{G} = \cup \tilde{G}_i^*$. A little algebra shows that:

$$P(\tilde{G}, G | \lambda(t)) = \prod_{i=1}^{|G|+1} \left( \exp\left( - \int_{G_{i-1}}^{G_i} dt \, (\Omega - \lambda(t) h(t - G_{i-1})) \right) \prod_{\tilde{g} \in \tilde{G}_i^*} (\Omega - \lambda(\tilde{g}) h(\tilde{g} - G_{i-1})) \right)$$

$$\times \prod_{i=1}^{|G|} \lambda(G_i) h(G_i - G_{i-1}) \prod_{i=1}^{|G|+1} \exp\left( - \int_{G_{i-1}}^{G_i} \lambda(t) h(t - G_{i-1}) dt \right) \quad (10)$$

$$= \Omega^{|G|+|\tilde{G}|} \exp\left(-\Omega T\right) \prod_{i=1}^{|G|} \left( \frac{\lambda(G_i) h(G_i - G_{i-1})}{\Omega} \right) \prod_{i=1}^{|G|+1} \prod_{\tilde{g} \in \tilde{G}_i^*} \left( 1 - \frac{\lambda(\tilde{g}) h(\tilde{g} - G_{i-1})}{\Omega} \right) \quad (11)$$

Comparing with equation (8), we have the following proposition:

**Proposition 3.** *The sets $(E, F)$ and $(G \cup \tilde{G}, G)$ are equivalent i.e. they have the same distribution.*

In other words, given a set of event times $G$, the inhomogeneous Poisson process-distributed points $\tilde{G}$ can be taken to be the events thinned in the procedure of section 3.2. The only complication left is that we do not know the function $\lambda(t)$ everywhere. This is easily overcome by uniformization (in fact, just by thinning, since we're dealing with a Poisson process). Specifically, let $G$ be the set of observed events and $\tilde{G}_{prev}$ the previous set of thinned events. To sample the new set $\tilde{G}_i^*$ from the Poisson process on $[G_{i-1}, G_i]$ with rate $(\Omega - \lambda(t)h(t - G_{i-1}))$, we first sample a set of points $A$ from a homogeneous Poisson process on $[G_{i-1}, G_i]$ with rate $\Omega$ and instantiate the Gaussian process on those points, *conditioned* on $G \cup \tilde{G}_{prev}$ and $l_{G \cup \tilde{G}_{prev}}$ (note that all this involves is conditionally sampling from a multivariate Gaussian[4]). Finally, we keep $a \in A$ with probability $1 - \frac{\lambda(a)h(a - G_{i-1})}{\Omega}$.

Having resampled $\tilde{G}$ (and the associated set of GP values), we next must resample the value of the GP at $G$. This does involve the sigmoid likelihood function, and we proceed by elliptical slice sampling [22][5]. Algorithm 1 lists the steps involved.

---

**Algorithm 1** Blocked Gibbs sampler for GP-modulated renewal process on the interval $[0, T]$

---

Input: Set of event times $G$, set of thinned times $\tilde{G}_{prev}$ and $l$ instantiated at $G \cup \tilde{G}_{prev}$.
Output: A new set of thinned times $\tilde{G}_{new}$ and a new instantiation $l_{G \cup \tilde{G}_{new}}$ of the GP on $G \cup \tilde{G}_{new}$.
 1: Sample $A \subset [0, T]$ from a Poisson process with rate $\Omega$.
 2: Sample $l_A | l_{G \cup \tilde{G}_{prev}}$.
 3: Thin $A$, keeping element $a \in A \cap [G_{i-1}, G_i]$ with probability $\left( 1 - \frac{\lambda^* \sigma(l(a))h(a - G_{i-1})}{\Omega} \right)$.
 4: Let $\tilde{G}_{new}$ be the resulting set and $l_{\tilde{G}_{new}}$ be the restriction of $l_A$ to this set. Discard $\tilde{G}_{prev}$ and $l_{\tilde{G}_{prev}}$.
 5: Resample $l_{G \cup \tilde{G}_{new}}$ using, for example, elliptical slice sampling.

---

The gamma prior on $\lambda^*$ is conjugate to the Poisson, resulting in a gamma posterior. We resampled the GP hyperparameters using slice sampling [23][5], while parameters of the hazard function were updated using Metropolis-Hastings moves along with equation (8).

## 4.1 Computational considerations

The inferential bottleneck in our model is the Gaussian process: sampling a GP on a set of points is, in the worst case, cubic in the size of that set. In our model, each iteration sees on average $|G| + 2|E|$ values of the GP, where $|G|$ is the number of observations and $|E|$ is the average number of points sampled from the subordinating Poisson process. Note that $|E|$ varies from iteration to iteration (being proportional to the scaling factor $\lambda^*$). Since we perform posterior inference on this quantity, the complexity of our model can be thought to adapt to that of the problem. This is in contrast with time-discretization approaches, where a resolution is picked beforehand, fixing the complexity of the inference problem accordingly. For instance, [2] use a resolution of 1ms to model neural spiking, making it impossible to naïvely deal with spike trains extending over more than a second. However as they demonstrate in their work, instantiating a GP on a regular lattice allows the development of fast *approximate* inference algorithms that scale linearly with the number of grid-points. In our case, the Gaussian processes is sampled at random locations. Moreover, these locations change each iteration, requiring the inversion of a new covariance matrix; this is the price we have to pay for an exact sampler.

One approach is to try reduce the number of thinned events $|E|$. Recall that our generative approach is to thin a sample from a subordinating, homogeneous Poisson process whose rate upper bounds the modulated hazard rate. We can reduce the number of thinned events by subordinating to an *inhomogeneous* Poisson process, one whose rate more closely resembles the instantaneous hazard rate. Thus, instead of using a single constant $\lambda^*$, one could use (say) a piecewise linear function

$\lambda^*(t)$ The more segments we use, the more flexibility we have; the price being the complexity of resampling this function, and slower mixing because of correlations it introduces.

This however does not help if $G$, the number of observations itself is large. In such a situation one has to call upon the vast literature concerning approximate inference for Gaussian processes [11]. The question then is how these approximation compare with those like [2]. We believe this is an interesting question in its own right, and raises the possibility of approximate inference algorithms that combine ideas from [2] with the adaptive nature of our approach.

## 5   Experiments

In this section we evaluate our model and sampler on a number of datasets. We used gamma distributed interevent times with shape parameter $\gamma \geq 1$. When $\gamma = 1$, we recover the Poisson process, and our model reduces to that of [6], while $\gamma > 1$ models 'refractoriness', where two events in quick succession are less likely than under a Poisson process. When appropriate, we place a noninformative prior on the shape parameter: an exponential with rate $0.1$ shifted to have a minimum value of 1. Note that for shape parameters less than 1, the renewal process becomes 'bursty' and the hazard function becomes unbounded. This is an interesting scenario but beyond the scope of this paper. An interesting issue concerns the identifiability of the shape parameter under our model. We find from our experiments that this is only a problem when the length scale of the intensity function is comparable to the refractory period of the renewal process. The base rate of the modulated renewal process (i.e. the rate when the intensity function is fixed at 1) is set to the empirical rate of the observed point process. As a result the identifiability of the shape parameter is a consequence of the dispersion of the point process rather than of some sort of rate matching.

**Synthetic data**. Our first set of experiments uses three synthetic datasets generated by modulating a gamma renewal process (shape parameter $\gamma = 3$) with three different functions (see figure 1):

- $\lambda_1(t) = 2\exp(t/5) + \exp(-((t-25)/10)^2), \quad t \in [0, 50]$: 44 events
- $\lambda_2(t) = 5\sin(t^2) + 6, \quad t \in [0, 5]$: 12 events
- $\lambda_3(t)$: a piecewise linear function , $\quad t \in [0, 100]$: 153 events

Additionally, for each function, we also generated 10 test sets. We ran three settings of our model: with the shape parameter fixed to 1 (MRP Exp), with the shape parameter fixed to the truth (MRP Gam3), and with a hyperprior on the shape parameter (MRP Full). For comparison, we also ran an approximate discrete-time sampler where the Gaussian process was instantiated on a regular grid covering the interval of interest. In this case, all intractable integrals were approximated numerically and we use elliptical slice sampling to run MCMC on this Gaussian vector.

Figure 1 shows the results from 5000 MCMC samples after a burn-in of 1000 samples. We quantify these in Table 1 by calculating the $l_2$ distance of the posterior means from the truth. We also calculated the mean predictive probabilities of the 10 test sequences. Not surprisingly, the inhomogeneous Poisson process forms a poor approximation to the gamma renewal process; it underestimates the intensity function required to produce a sequence of events with refractory intervals. Fixing the shape parameter to the truth significantly reduces the $l_2$ error and increases the predictive probabilities, but interestingly, for these datasets, the model with a prior on the shape parameter performs comparably with the 'oracle' model. We have also included plots of the posterior distribution over the gamma parameter; these are peaked around 3. Discretizing time into a 100 bins (Disc100) results in comparable performance for the first two datasets on the $l_2$ error; for the third, (which spans a longer interval and has a larger event count), we had to increase the resolution to 500 bins to improve accuracy. Discretizing to 25 bins was never sufficient. A conclusion is that with time discretization, for a small bias, one must be conservative in choosing the time-resolution; however, evaluating a GP on a fine grid can result in slow mixing. Our sampler has the advantage of automatically picking the 'right' resolution. However as we discussed in the section on computation, time discretization has its own advantages that make it a viable model [2].

**Coal mine disaster data**. For our next experiment, we ran our model on the coal mine disaster dataset commonly used in the point process literature. This dataset records the dates of a series of 191 coal mining disasters, each of which killed ten or more men [24]. Figure 2(left) shows the posterior mean of the intensity function (surrounded by 1 standard deviation) returned by our model. Not included is the posterior distribution over the shape parameter; this concentrated in the interval 1 to around 1.1, suggesting that the data is well modelled as an inhomogeneous Poisson process, and

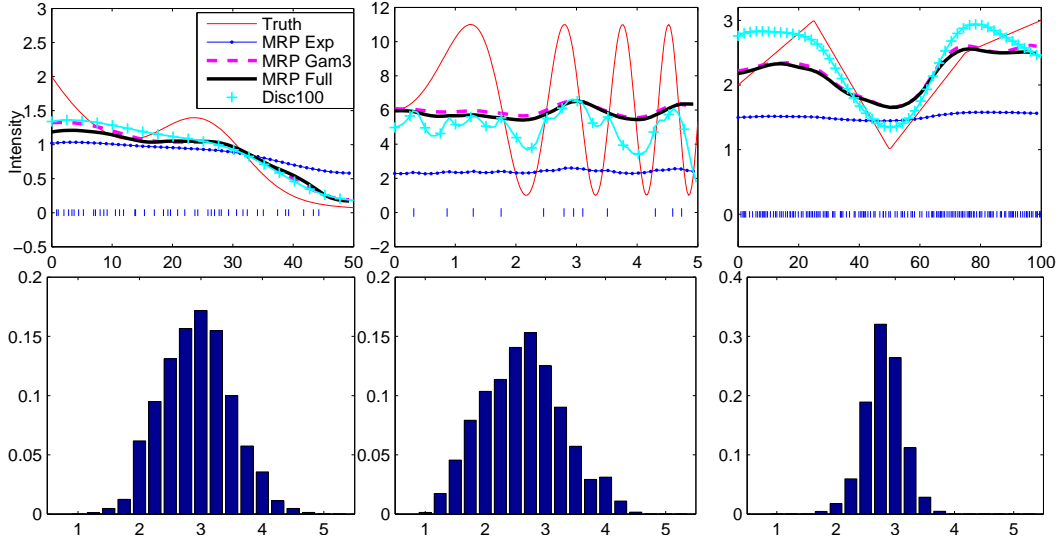

Figure 1: Synthetic Datasets 1-3: Posterior mean intensities plotted against time (top) and gamma shape posteriors (bottom)

|  | MRP Exp | MRP Gam3 | MRP Full | Disc25 | Disc100 |
|---|---|---|---|---|---|
| $l_2$ error | 7.8458 | 3.19 | 2.548 | 4.089003 | **2.426973** |
| log pred. prob. | -47.5469 | -38.0703 | **-37.3712** | -41.646350 | -41.016425 |
| $l_2$ error | 141.0067 | **56.2183** | 58.4361 | 91.321069 | 57.896300 |
| log pred. prob. | -3.704396 | **-2.945298** | -3.280871 | -5.245478 | -3.848443 |
| $l_2$ error | 82.0289 | **11.4167** | 13.4441 | 122.335151 | 38.047332 |
| log pred. prob. | -89.8787 | **-48.2777** | -48.57 | 87.170034 | -55.802997 |

Table 1: $l_2$ distance from the truth and mean log-predictive probabilities of the held-out datasets for synthetic datasets 1(top) to 3(bottom).

is in agreement with [24]. As sanity check, and to shed further light on the issue of identifiability, we processed the dataset by deleting every alternate event. With such a transformation, a homogeneous Poisson would reduce to a gamma renewal process with shape 2. Our model returns a posterior peaked around 1.5 (in agreement with the form of the inhomogeneity). Note that the posteriors over intensity functions are similar (except for the obvious scaling factor of about 2).

**Spike timing data** We next ran our model on neural spike train data recorded from grasshopper auditory receptor cells [25]. This dataset is characterized by a relatively high firing rate ($\sim$ 150

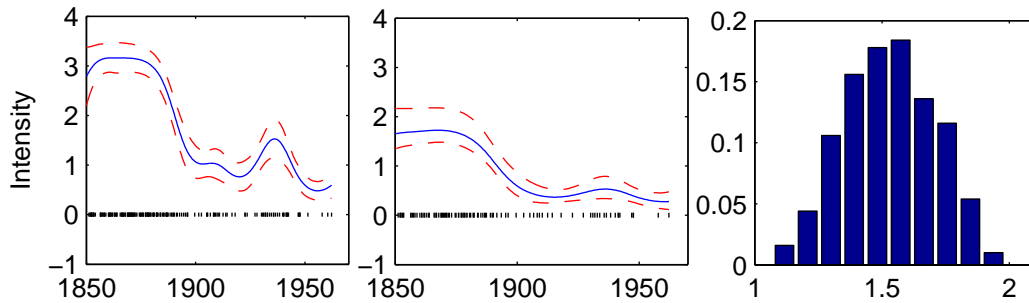

Figure 2: Left: Posterior mean intensity for coal mine data with 1 standard deviation error bars (plotted against time in years). Centre: Posterior mean intensity for 'thinned' coalmine data with 1 standard deviation error bars. Right: Gamma shape posterior for 'thinned' coal mine data.

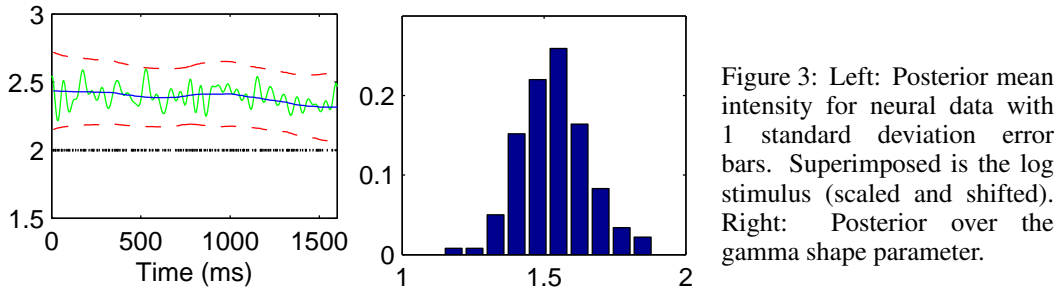

Figure 3: Left: Posterior mean intensity for neural data with 1 standard deviation error bars. Superimposed is the log stimulus (scaled and shifted). Right: Posterior over the gamma shape parameter.

| | Synthetic dataset 1 | | | Coalmine dataset | | |
|---|---|---|---|---|---|---|
| | Mean ESS | Minimum ESS | Time(sec) | Mean ESS | Minimum ESS | Time(sec) |
| Gibbs | $93.45 \pm 6.91$ | $50.94 \pm 5.21$ | 77.85 | $53.54 \pm 8.15$ | $24.87 \pm 7.38$ | 282.72 |
| MH | $56.37 \pm 10.30$ | $19.34 \pm 11.55$ | 345.44 | $47.83 \pm 9.18$ | $18.91 \pm 6.45$ | 1703 |

Table 2: Sampler comparisons. Numbers are per 1000 samples.

Hz), making refractory effects more prominent. We plot the posterior distribution over the intensity function given a sequence of 200 spikes in a 1.6 second interval. We also included the posterior distribution over gamma shape parameters in figure 3; this concentrates around 1.5, agreeing with the refractory nature of neuronal firing. The results above follow from using noninformative hyperpriors; we have also plotted the log-transformed stimulus, an amplitude-modulated signal. In practice, other available knowledge (viz. the shape parameter, the stimulus length-scale, the transformation from the stimulus to the input of the neuron etc) can be used to make more accurate inferences.

**Computational efficiency and mixing**. For our final experiment, we compare our proposed blocked Gibbs sampler with the Metropolis-Hastings sampler of [6]. We ran both algorithms on two datasets, synthetic dataset 1 from section 5 and the coal mine disaster dataset. All involved 20 MCMC runs with 5000 iterations each (following a burn-in period of a 1000 iterations). For both datasets, we evaluated the latent GP on a uniform grid of 200 points, calculating the effective sample size (ESS) of each component of the Gaussian vectors (using R-CODA [26]). For each run, we return the mean and the minimum ESS across all 200 components. In Table 2, we report these numbers: not only does our sampler mix faster (resulting in larger ESSs), but also takes less computation time. Additionally, our sampler is simpler and more natural to the problem, and does not require any external tuning.

# 6 Discussion

We have described how to produce exact samples from a nonstationary renewal process whose hazard function is modulated by a Gaussian process. Our scheme is based on the idea of uniformization, and using this idea, we also develop a novel MCMC sampler. There are a number of interesting avenues worth following. First is the restriction that the hazard function be bounded: while this covers a large and useful class of renewal processes, it is worth considering how our approach can be extended to produce exact or approximate samples for renewal processes with unbounded hazard functions. In any case, following [13], it is easy to extend our ideas to Bayesian inference for more general point processes. Because of the latent Gaussian process, our approach will not scale well to large problems; however there is a vast literature concerning approximate sampling for Gaussian processes. An important question is how these approximations compare to approximations introduced via time-discretization. Finally, even though we considered GP modulating functions, our uniformization-based sampler will also be useful for Bayesian inference involving simpler priors on modulating functions, eg. splines or Markov jump processes.

**Acknowledgements**

We thank the Gatsby Charitable Foundation for generous funding. We thank Ryan Adams and Iain Murray for code and comments; and Jakob Macke and Lars Buesing for useful discussions. The grasshopper data was collected by Ariel Rokem at Andreas Herz's lab and provided through the CRCNS program (http://crcns.org).

## Footnotes

[1]With renewal processes there is an ambiguity about the time of the first event, which is typically taken to be exponentially distributed. It is straightforward to handle this case.

[2]We parametrize the hazard function to produce 1 event per unit time; other parametrizations may be used.

[3]Recall that $G_0 = 0$. We also take $G_{|G|+1} = T$.

[4]In particular, it does not require any sophisticated GP sampling algorithm

[5]Code available on Iain Murray's website: http://homepages.inf.ed.ac.uk/imurray2/

# References

[1] J. F. Lawless and K. Thiagarajah. A point-process model incorporating renewals and time trends, with application to repairable systems. *Technometrics*, 38(2):131–138, 1996.

[2] John P. Cunningham, Byron M. Yu, Krishna V. Shenoy, and Maneesh Sahani. Inferring neural firing rates from spike trains using Gaussian processes. In *Advances in Neural Information Processing Systems 20*, 2008.

[3] T. Parsons. Earthquake recurrence on the south Hayward fault is most consistent with a time dependent, renewal process. *Geophysical Research Letters*, 35, 2008.

[4] V. Paxson and S. Floyd. Wide area traffic: the failure of Poisson modeling. *IEEE/ACM Transactions on Networking*, 3(3):226–244, June 1995.

[5] C. Wu. Counting your customers: Compounding customer's in-store decisions, interpurchase time and repurchasing behavior. *European Journal of Operational Research*, 127(1):109–119, November 2000.

[6] Ryan P. Adams, Iain Murray, and David J. C. MacKay. Tractable nonparametric Bayesian inference in Poisson processes with Gaussian process intensities. In *Proceedings of the 26th International Conference on Machine Learning (ICML)*, 2009.

[7] A. Jensen. Markoff chains as an aid in the study of Markoff processes. *Skand. Aktuarietiedskr.*, 36:87–91, 1953.

[8] V. Rao and Y. W. Teh. Fast MCMC sampling for Markov jump processes and continuous time Bayesian networks. In *Proceedings of the International Conference on Uncertainty in Artificial Intelligence*, 2011.

[9] D.R. Cox. The statistical analysis of dependencies in point processes. In P.A. Lewis, editor, *Stochastic point processes*, pages 55–56. New York: Wiley 1972, 1972.

[10] Robert E. Kass and Valérie Ventura. A spike-train probability model. *Neural Computation*, 13(8):1713–1720, 2001.

[11] C. E. Rasmussen and C. K. I. Williams. *Gaussian Processes for Machine Learning*. MIT Press, 2006.

[12] M. Berman. Inhomogeneous and modulated gamma processes. *Biometrika*, 68(1):143, 1981.

[13] Yosihiko Ogata. On Lewis' simulation method for point processes. *IEEE Transactions on Information Theory*, 27(1):23–31, 1981.

[14] Mark Berman and T. Rolf Turner. Approximating point process likelihoods with GLIM. *Journal of the Royal Statistical Society. Series C (Applied Statistics)*, 41(1):pp. 31–38, 1992.

[15] I. Sahin. A generalization of renewal processes. *Operations Research Letters*, 13(4):259–263, May 1993.

[16] Emery N. Brown, Riccardo Barbieri, Valérie Ventura, Robert E. Kass, and Loren M. Frank. The time-rescaling theorem and its application to neural spike train data analysis. *Neural computation*, 14(2):325–46, February 2002.

[17] I. Gerhardt and B. L. Nelson. Transforming renewal processes for simulation of nonstationary arrival processes. *INFORMS Journal on Computing*, 21(4):630–640, April 2009.

[18] Bo Henry Lindqvist. Nonparametric estimation of time trend for repairable systems data. In V.V. Rykov, N. Balakrishnan, and M.S. Nikulin, editors, *Mathematical and Statistical Models and Methods in Reliability*, Statistics for Industry and Technology, pages 277–288. Birkhuser Boston, 2011.

[19] P. A. W. Lewis and G. S. Shedler. Simulation of nonhomogeneous Poisson processes with degree-two exponential polynomial rate function. *Operations Research*, 27(5):1026–1040, September 1979.

[20] J. F. C. Kingman. *Poisson processes*, volume 3 of *Oxford Studies in Probability*. The Clarendon Press Oxford University Press, New York, 1993. Oxford Science Publications.

[21] J George Shanthikumar. Uniformization and hybrid simulation/analytic models of renewal processes. *Oper. Res.*, 34:573–580, July 1986.

[22] Iain Murray, Ryan Prescott Adams, and David J.C. MacKay. Elliptical slice sampling. *JMLR: W&CP*, 9, 2010.

[23] Iain Murray and Ryan Prescott Adams. Slice sampling covariance hyperparameters of latent Gaussian models. In *Advances in Neural Information Processing Systems 23*, 2010.

[24] B. Y. R. G. Jarrett. A note on the intervals between coal-mining disasters. *Biometrika*, 66(1):191–193, 1979.

[25] Ariel Rokem, Sebastian Watzl, Tim Gollisch, Martin Stemmler, and Andreas V.M. Herz. Spike-Timing Precision Underlies the Coding Efficiency of Auditory Receptor Neurons. *Journal of Neurophysiology*, pages 2541–2552, 2006.

[26] Martyn Plummer, Nicky Best, Kate Cowles, and Karen Vines. CODA: Convergence diagnosis and output analysis for MCMC. *R News*, 6(1):7–11, March 2006.

